# Development and Regeneration of Eye-Brain Maps: A Computational Model

**J.D. Cowan** and **A.E. Friedman**
Department of Mathematics, Committee on
Neurobiology, and Brain Research Institute,
The University of Chicago, 5734  S. Univ. Ave.,
Chicago, Illinois 60637

## ABSTRACT

We outline a computational model  of the development and regeneration of specific eye-brain circuits. The model comprises a self-organizing map-forming network which uses local Hebb rules, constrained by molecular markers.  Various simulations of the development of eye-brain maps in fish and frogs are described.

## 1 INTRODUCTION

The brain is a biological computer of immense complexity comprising highly specialized neurons and neural circuits.  Such neurons are interconnected with high *specificity*  in many regions of the brain, if not in all. There are also many observations which indicate that there is also considerable circuit *plasticity*.  Both specificity and plasticity are found in the development and regeneration of eye-brain connections in vertebrates.  Sperry (1944) first demonstrated specificity in the regeneration of eye-brain connections in frogs following optic nerve section and eye rotation; and Gaze and Sharma (1970) and Yoon (1972) found evidence for plasticity in the expanded and compressed *maps* which regenerate following eye and brain lesions in goldfish.  There are now many experiments which indicate that the formation of connections involves both specificity and plasticity.

## 1.1 EYE-BRAIN MAPS AND MODELS

Fig. 1 shows the retinal map found in the optic lobe or *tectum* of fish and frog. The map is topological, i.e.; neighborhood relationships in the retina are preserved in the optic tectum. How does such a map develop? Initially there is considerable disorder in the

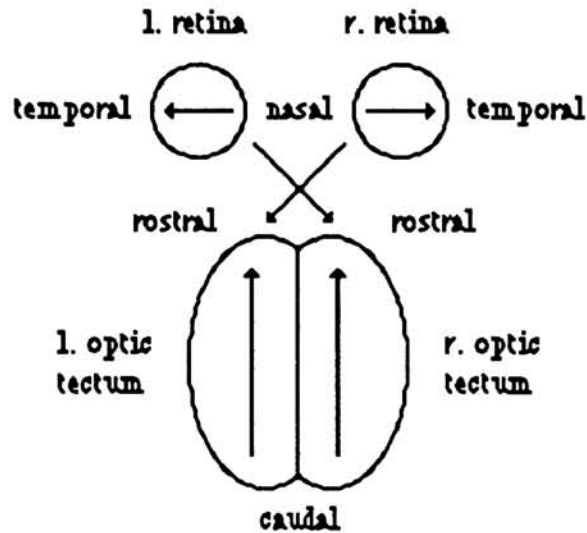

**Figure 1:** The normal retino-tectal map in fish and frog. Temporal retina projects to (contralateral) rostral tectum; nasal retina to (contralateral) caudal tectum.

pathway: retinal ganglion cells make contacts with many widely dispersed tectal neurons. However the mature pathway shows a high degree of topological order. How is such an organized map achieved? One answer was provided by Prestige & Willshaw (1975): retinal axons and tectal neurons are *polarized* by contact adhesion molecules distributed such that axons from one end of the retina are stickier than those from the other end, and neurons at one end of the tectum are (correspondingly) stickier than those at the other end. Of course this means that isolated retinal axons will all tend to stick to one end of the tectum. However if such axons *compete* with each other for tectal terminal sites (and if tectal sites compete for retinal axon terminals), less sticky axons will be displaced, and eventually a topological map will form. The Prestige-Willshaw theory explains many observations indicating neural specificity. It does not provide for plasticity: the ability of retino-tectal systems to adapt to changed target conditions, and vice-versa. Willshaw and von der Malsburg (1976, 1977) provided a theory for the plasticity of map reorganization, by postulating the synaptic growth in development is Hebbian. Such a mechanism provides self-organizing properties in retino-tectal map formation and reorganization. Whitelaw & Cowan (1981) combined both sticky molecules and Hebbian synaptic growth to provide a theory which explains both the specificity and plasticity of map formation and reorganization in a reasonable fashion.

There are many experiments, however, which indicate that such theories are too simple. Schmidt & Easter (1978) and Meyer (1982) have shown that retinal axons interact with

each other in a way which influences map formation. It is our view that there are (probably) at least two different types of sticky molecules in the system: those described above which mediate retino-tectal interactions, and an additional class which mediates axo-axonal interactions in a different way. In what follows we describe a model which incorporates such interactions. Some aspects of our model are similar to those introduced by Willshaw & von der Malsburg (1979) and Fraser (1980). Our model can simulate almost all experiments in the literature, and provides a way to titrate the relative strenghts of intrinsic *polarity* markers mediating retino-tectal interactions, (postulated) *positional* markers mediating axo-axonal interactions, and stimulus-driven Hebbian synaptic changes.

## 2 MODELS OF MAP FORMATION AND REGENERATION

### 2.1. THE WHITELAW-COWAN MODEL

Let $s_{ij}$ be the strength or weight of the synapse made by the ith retinal axon with the jth tectal cell. Then the following differential equation expresses the changes in $s_{ij}$:

$$\dot{s}_{ij} = c_{ij} (r_i - \alpha) t_j - \tfrac{1}{2} (N_r^{-1} \textstyle\sum_i + N_t^{-1} \sum_j )(c_{ij} (r_i - \alpha) t_j) \qquad (1)$$

where $N_r$ is the number of retinal ganglion cells and $N_t$ the number of tectal neurons, $c_{ij}$ is the "stickiness" of the ijth contact, $r_i$ denotes retinal activity and $t_j = \sum_i s_{ij} r_i$ is the corresponding tectal activity, and $\alpha$ is a constant measuring the rate of receptor destabilization (see Whitelaw & Cowan (1981) for details). In addition both retinal and tectal elements have fixed lateral inhibitory contacts. The dynamics described by eqn.1 is such that both $\Sigma_i s_{ij}$ and $\Sigma_j s_{ij}$ tend to constant values T and R respectively, where T is the total amount of tectal receptor material available per neuron, and R is the total amount of axonal material available per retinal ganglion cell: thus if sij increases anywhere in the net, other synapses made by the ith axon will decrease, as will other synapses on the jth tectal neuron. In the current terminology, this process is referred to as "winner-take-all".

For purposes of illustration consider the problem of connecting a line of $N_r$ retinal ganglion cells to a line of $N_t$ tectal cells. The resulting maps can then be represented by two-dimensional matrices, in which the area of the square at the ijth intersection represents the weight of the synapse between the ith retinal axon and the jth tectal cell. The normal retino-tectal map is represented by large squares along the matrix diagonal., (see Whitelaw & Cowan (1981) for terminology and further details). It is fairly obvious that the only solutions to eqn. (1) lie along the matrix diagonal, or the anti-diagonal, as shown in fig. 2. These solutions correspond, respectively,  to normal  and inverted topological maps. It follows that if the affinity $c_{ij}$ of the ith retinal ganglion cell for the jth tectal neuron is constant, a map will form consisting of normal and inverted local patches. To obtain a globally normal map it is necessary to *bias* the system. One way to do this is to suppose that $c_{ij} = \xi a_i a_j$, where $a_i$ and $a_j$ are respectively, the concentrations

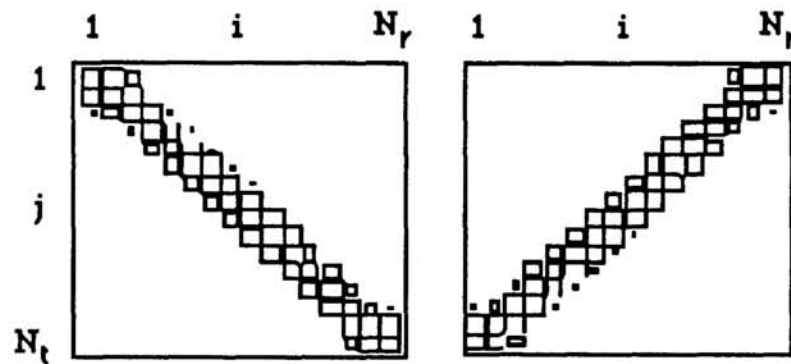

**Figure 2:** Diagonal and anti-diagonal solutions to eqn.1.  Such solutions correspond, respectively, to normal and inverted maps.

of sticky molecules on the tips of retinal axons and on the surfaces of tectal neurons, and $\xi$ is a constant.  A good candidate for such a molecule is the recently discovered toponymic or TOP molecule found in chick retina and tectum (Trisler & Collins, 1987).  If $a_i$ and $a_j$ are distributed in the *graded* fashion shown in fig. 3,  then  the system is biased in favor of the normally oriented map.

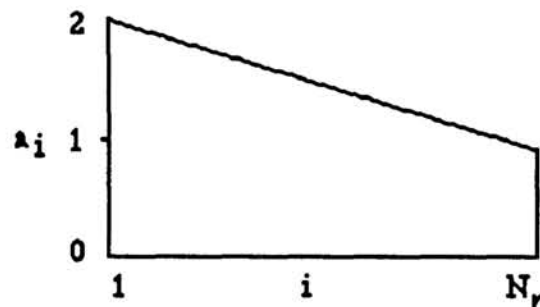

**Figure 3:** Postulated distribution of sticky molecules in the retina.  A similar distribution is supposed to exist in the tectum.

## 2.2 INADEQUACIES

The Whitelaw-Cowan model simulates the normal development of monocular retinotectal maps, starting from either diffuse or scrambled initial maps, or from no map.  In addition it simulates the compressed, expanded, translocated, mismatched and rotated maps which have been described in a variety of surgical contexts.  However it fails in the following respects: a. Although tetrodotoxin (TTX) blocks the refinement of retinotopic maps in salamanders, a coarse map can still develop in the absence of retinal activity Harris (1980).  The model will not simulate this effect.  b. Although the model  simulates the formation of double maps in "classical" compound eyes {made from a half-left and a half right eye} (Gaze, Jacobson, & Szekely, 1963), it fails to account for the reprogramming observed in "new" compound eyes {made by cutting a slit down the middle of a tadpole eye} (Hunt & Jacobson, 1974), and fails to simulate the forming of a

normal retinotopic map to a compound tectum {made from two posterior halves} (Sharma, 1975).

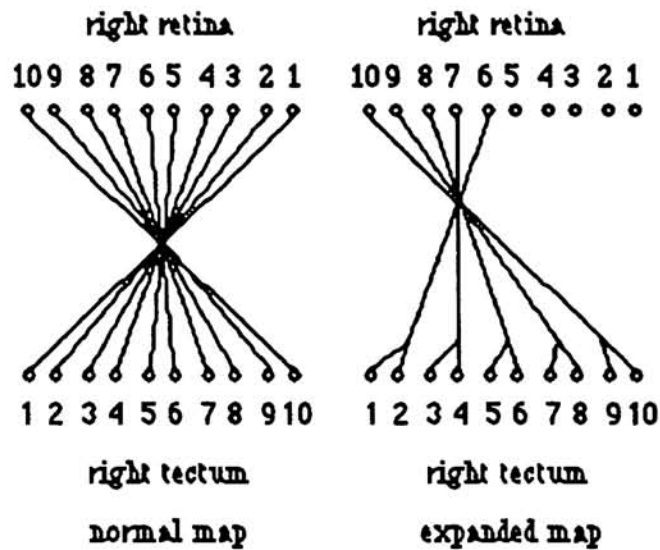

Figure 4: The normal and expanded maps which form after the prior expansion of axons from a contralateral half-eye. The two maps are actually superposed, but for ease of exposition are shown separately.

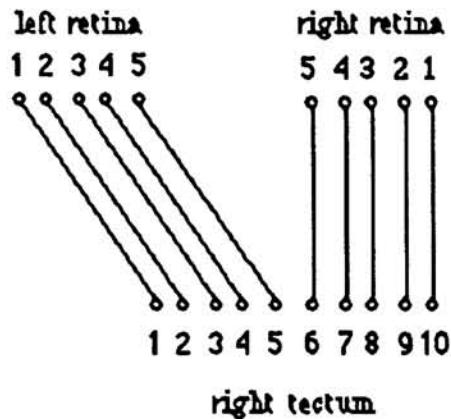

Figure 5: Results of Meyer's experiment. Fibers from the right half-retina fail to contact their normal targets and instead make contact with available targets, but with reversed polarity.

c. More significantly, it fails to account for the apparent retinal *induction* reported by Schmidt, Cicerone & Easter (1978) in which following the expansion of retinal axons from a goldfish half-eye over an entire (contralateral) tectum, and subsequent sectioning of the axons, diverted retinal axons from the other (intact) eye are found to expand over the tectum, as if they were also from a half-eye. This has been interpreted to imply that the tectum has no intrinsic markers, and that all its markers come from the retina (Chung & Cooke, 1978). However Schmidt et.al. also found that the diverted axons also map normally. Fig. 4 shows the result. d. There is also an important mismatch experiment

carried out by Meyer (1979) which the model cannot simulate. In this experiment the left half of an eye and its attached retinal axons are surgically removed, leaving an intact normal half-eye map. At the same time the right half the other eye and its attached axons are removed, and the axons from the remaining half eye are allowed to innervate the tectum with the left-half eye map. The result is shown in fig. 5. e. Finally, there are now a variety of chemical assays of the nature of the affinities which retinal axons have for each other, and for tectal target sites. Thus Bonhoffer and Huff (1980) found that growing retinal axons stick preferentially to rostral tectum. This is consistent with the model. However, using a different assay Halfter, Claviez & Schwarz (1981) also found that tectal fragments tend to stick preferentially to that part of the retina which corresponds to caudal tectum, i.e.; to *nasal* retina. This appears to contradict the model, and the first assay.

## 3 A NEW MODEL FOR MAP FORMATION

The Whitelaw-Cowan model can be modified and extended to replicate much of the data described above. The first modification is to replace eqn.1 by a more nonlinear equation. The reason for this is that the above equation has no threshold below which contacts cannot get established. In practice Whitelaw and I modified the equations to incorporate a small threshold effect. Another way is to make synaptic growth and decay exponential rather than linear. An equation expressing this can be easily formulated, which also incorporates *axo-axonal interactions*, presumed to be produced by neural contact adhesion molecules (nCAM) of the sort discovered by Edelman (1983) which seem to mediate the axo-axonal adhesion observed in tissue cultures by Boenhoffer & Huff (1985). The resulting equations take the form:

$$\dot{s}_{ij} = \lambda_j + c_{ij} [\mu_{ij} + (r_i - \alpha)t_j] s_{ij}$$
$$- \tfrac{1}{2} s_{ij} (T^{-1}\textstyle\sum_i + R^{-1}\textstyle\sum_j )\{\lambda_j + c_{ij} [\mu_{ij} + (r_i - \alpha)t_j] s_{ij}\} \qquad (2)$$

where $\lambda_j$ represents a general nonspecific growth of retinotectal contacts, presumed to be controlled and modulated by nerve growth factor (Campenot, 1982). The main difference between eqns. 1 and 2 however, lies in the coefficients $c_{ij}$. In eqn. 1, $c_{ij} = \xi a_i a_j$. In eqn. 2, $c_{ij}$ expresses several different effects: (a). Instead of just one molecular species on the tips of retinal axons and on corresponding tectal cell surfaces, as in eqn.1, two molecular species or two states of one species can be postulated to exist on these sites. In such a case the term $\xi a_i a_j$ is replaced by $\sum \xi_{ab} a_i b_j$ where a and b are the different species, and the sum is over all possible combinations aa, ab etc. A number of possibilities exist in the choice of $\xi_{ab}$. One possibility that is consistent with most of the biochemical assays described earlier is $\xi_{aa} = \xi_{bb} < \xi_{ab} = \xi_{ba}$ in which each species prefers the other, the so-called heterophilic case. (b) The mismatch experiment cited earlier (Meyer, 1979) indicates that existing axon projections tend to exclude other axons, especially inappropriate ones, from innervating occupied areas. One way to incorporate such geometric effects is to suppose that each axon which establishes contact with a tectal neuron *occludes* tectal markers there by a factor proportional to its synaptic

weight $s_{ij}$. Thus we subtract from the coefficient $c_{ij}$ a fraction proportional to $T^{-1}\sum'_k s_{kj}$ where $\sum'_k$ means $\sum_{k \neq i}$. (c) The mismatch experiment also indicates that map formation depends in part on a tendency for axons to stick to their retinal neighbors, in addition to their tendency to stick to tectal cell surfaces. We therefore append to $c_{ij}$ the term $\sum'_k \bar{s}_{kj} f_{ik}$ where $s_{kj}$ is a local average of $s_{kj}$ and its nearest tectal neighbors, and where $f_{ik}$ measures the mutual stickiness of the ith and kth retinal axons: non-zero only for nearest *retinal* neighbors. {Again we suppose this stickiness is produced by the interaction of two molecular species etc.; specifically the neuronal CAMs discovered by Edelman, but we do not go into the details}. (d) With the introduction of occlusion effects and axo-axonal interactions, it becomes apparent that *debris* in the form of degenerating axon fragments adhering to tectal cells, following optic nerve sectioning, can also influence map formation. Incoming nerve axons can stick to debris, and debris can occlude markers. There are in fact four possibilities: debris can occlude tectal markers, markers on other debris, or on incoming axons; and incoming axons can occlude markers on debris. All these possibilities can be included in the dependence of $c_{ij}$ on $s_{ij}$, $s_{kj}$ etc.

The model which results from all these modifications and extensions is much more complex in its mathematical structure than any of the previous models. However computer simulation studies show it to be capable of correctly reproducing the observed details of almost all the experiments cited above. Fig. 6, for example shows a simulation of the retinal "induction" experiments of Schmidt *et.al.*

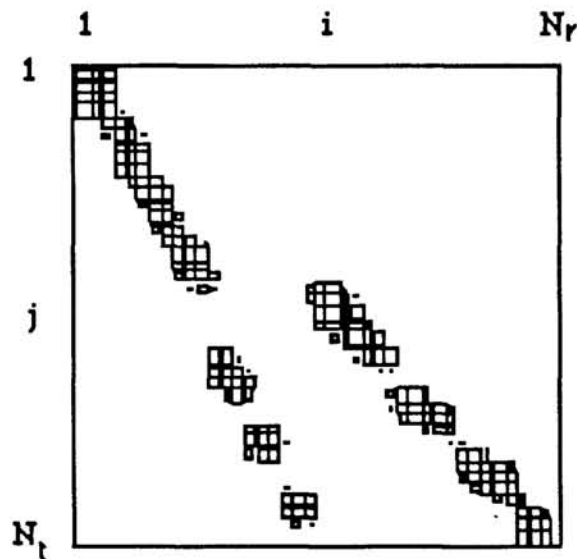

**Figure 6:**   Simulation of the Schmidt *et.al.* retinal induction experiment. A nearly normal map is intercalated into an expanded map.

This simulation generated both a patchy expanded and a patchy nearly normal map. These effects occur because some incoming retinal axons stick to debris left over from

the previous expanded map, and other axons stick to non-occluded tectal markers. The axo-axonal positional markers control the formation of the expanded map, whereas the retino-tectal polarity markers control the formation of the nearly normal map.

# 4 CONCLUSIONS

The model we have outlined combines Hebbian plasticity with intrinsic, genetic eye-brain and axo-axonic markers, to generate correctly oriented retinotopic maps. It permits the simulation of a large number of experiments, and provides a consistent explanation of almost all of them. In particular it shows how the apparent induction of central markers by peripheral effects, as seen in the Schmidt-Cicerone-Easter experiment (Schmidt *et.al.* 1978), can be produced by the effects of debris; and the polarity reversal seen in Meyer's experiment (Meyer 1979), can be produced by axo-axonal interactions.

## Acknowledgements

We thank the System Development Foundation, Palo Alto, California, and The University of Chicago Brain Research Foundation for partial support of this work.

## References

Boenhoffer, F. & Huf, J. (1980), Nature, **288**, 162-164.; (1985), Nature, **315**, 409-411.
Campenot, R.B. (1982), Develop. Biol., **93**, 1.
Chung, S.-H. & Cooke, J.E. (1978), Proc. Roy. Soc. Lond. B *201*, 335-373.
Edelman, G.M., (1983), Science, **219**, 450-454.
Fraser, S. (1980), Develop. Biol., **79**, 453-464.
Gaze, R.M. & Sharma, S.C. (1970), Exp. Brain Res., 10, 171-181.
Gaze, R.M., Jacobson, M. & Szekely, T. (1963), J. Physiol. (Lond.), **165**, 484-499.
Halfter, W., Claviez, M. & Schwarz, U. (1981), Nature, **292**, 67- 70.
Harris, W.A. (1980), J. Comp. Neurol., **194**, 303-323.
Hubel, D.H. & Wiesel, T.N. (1974), J. Comp. Neurol. 158, 295-306.
Hunt, R.K. & Jacobson. M. (1974), Devel. Biol. 40, 1-15.
Malsburg, Ch.v.d. & Willshaw, D.J. (1977), PNAS, **74**, 5176-5178.
Meyer, R.L. (1979), Science, **205**, 819-821; (1982), Curr. Top. Develop. Biol., **17**, 101-145.
Prestige, M. & Willshaw, D.J. (1975), Proc. Roy. Soc. B, 190, 77-98.
Schmidt, J.T. & Easter, S.S. (1978), Exp. Brain Res., **31**, 155-162.
Schmidt, J.T., Cicerone, C.M. & Easter, S.S. (1978), J. Comp. Neurol., **177**, 257-288.
Sharma, S.C. (1975), Brain Res., **93**, 497-501.
Sperry, R.W. (1944), J. Neurophysiol., **7**, 57-69.
Trisler, D. & Collins, F. (1987), Science, **237**, 1208-1210.
Whitelaw, V.A. & Cowan, J.D. (1981), J. Neurosci., **1**, *12*, 1369-1387.
Willshaw, D.J. & Malsburg, Ch.v.d. (1976), Proc. Roy. Soc. B, 194, 431-445; (1979), Phil. Trans. Roy. Soc. (Lond.), **B**, *287*, 203-254.
Yoon, M. (1972), Amer. Zool., **12**, 106.